# Neural Computing with Small Weights

**Kai-Yeung Siu**
Dept. of Electrical & Computer Engineering
University of California, Irvine
Irvine, CA 92717

**Jehoshua Bruck**
IBM Research Division
Almaden Research Center
San Jose, CA 95120-6099

## Abstract

An important issue in neural computation is the dynamic range of weights in the neural networks. Many experimental results on learning indicate that the weights in the networks can grow prohibitively large with the size of the inputs. Here we address this issue by studying the tradeoffs between the depth and the size of weights in polynomial-size networks of linear threshold elements (LTEs). We show that there is an efficient way of simulating a network of LTEs with large weights by a network of LTEs with small weights. In particular, we prove that every depth-$d$, polynomial-size network of LTEs with *exponentially large* integer weights can be simulated by a depth-$(2d + 1)$, polynomial-size network of LTEs with *polynomially bounded* integer weights. To prove these results, we use tools from harmonic analysis of Boolean functions. Our technique is quite general, it provides insights to some other problems. For example, we are able to improve the best known results on the depth of a network of linear threshold elements that computes the $COMPARISON$, $SUM$ and $PRODUCT$ of two $n$-bits numbers, and the $MAXIMUM$ and the $SORTING$ of $n$ $n$-bit numbers.

## 1 Introduction

The motivation for this work comes from the area of neural networks, where a linear threshold element is the basic processing element. Many experimental results on learning have indicated that the magnitudes of the coefficients in the threshold elements grow very fast with the size of the inputs and therefore limit the practical use of the network. One natural question to ask is the following: How limited

is the computational power of the network if we restrict ourselves to threshold elements with only "small" growth in the coefficients? We answer this question by showing that we can trade-off an exponential growth with a polynomial growth in the magnitudes of coefficients by increasing the depth of the network by a factor of almost two and a polynomial growth in the size.

**Linear Threshold Functions:** A linear threshold function $f(X)$ is a Boolean function such that

$$f(X) = sgn(F(X)) = \begin{cases} 1 & \text{if } F(X) > 0 \\ -1 & \text{if } F(X) < 0 \end{cases}$$

where

$$F(X) = \sum_{i=1}^{n} w_i \cdot x_i + w_0$$

Throughout this paper, a *Boolean function* will be defined as $f : \{1, -1\}^n \to \{1, -1\}$; namely, 0 and 1 are represented by 1 and $-1$, respectively. Without loss of generality, we can assume $F(X) \neq 0$ for all $X \in \{1, -1\}^n$. The coefficients $w_i$ are commonly referred to as the *weights* of the threshold function. We denote the class of all linear threshold functions by $LT_1$.

$\widehat{LT}_1$ **functions:** In this paper, we shall study a subclass of $LT_1$ which we denote by $\widehat{LT}_1$. Each function $f(X) = sgn(\sum_{i=1}^{n} w_i \cdot x_i + w_0)$ in $\widehat{LT}_1$ is characterized by the property that the weights $w_i$ are integers and bounded by a polynomial in $n$, i.e. $|w_i| \leq n^c$ for some constant $c > 0$.

**Threshold Circuits:** A *threshold circuit* [5, 10] is a Boolean network in which every gate computes an $\widehat{LT}_1$ function. The *size* of a threshold circuit is the number of $\widehat{LT}_1$ elements in the circuit. Let $\widehat{LT}_k$ denote the class of threshold circuits of *depth k* with the *size bounded by a polynomial* in the number of inputs. We define $LT_k$ similarly except that we allow each gate in $LT_k$ to compute an $LT_1$ function.

Although the definition of $(LT_1)$ linear threshold function allows the weights to be real numbers, it is known [12] that we can replace each of the real weights by integers of $O(n \log n)$ bits, where $n$ is the number of input Boolean variables. So in the rest of the paper, we shall assume without loss of generality that all weights are integers. However, this still allows the magnitudes of the weights to increase exponentially fast with the size of the inputs. It is natural to ask if this is necessary. In other words, is there a linear threshold function that must require exponentially large weights? Since there are $2^{\Omega(n^2)}$ linear threshold functions in $n$ variables [8, 14, 15], there exists at least one which requires $\Omega(n^2)$ bits to specify the weights. By the pigeonhole principle, at least one weight of such a function must need $\Omega(n)$ bits, and thus is exponentially large in magnitude. i.e.

$$\widehat{LT}_1 \subsetneq LT_1$$

The above result was proved in [9] using a different method by explicitly constructing an $LT_1$ function and proving that it is not in $\widehat{LT}_1$. In the following section, we shall show that the $COMPARISON$ function (to be defined later) also requires exponentially large weights. We will refer to this function later on in the proof of

our main results. **Main Results:** The fact that we can simulate a linear threshold function with exponentially large weights in a 'constant' number of layers of elements with 'small' weights follows from the results in [3] and [11]. Their results showed that the sum of $n$ $n$-bit numbers is computable in a constant number of layers of 'counting' gates, which in turn can be simulated by a constant number of layers of threshold elements with 'small' weights. However, it was not explicitly stated how many layers are needed in each step of their construction and direct application of their results would yield a constant such as 13. In this paper, we shall reduce the constant to 3 by giving a more 'depth'-efficient algorithm and by using harmonic analysis of Boolean functions [1, 2, 6]. We then generalize this result to higher depth circuits and show how to simulate a threshold circuit of depth-$d$ and exponentially large weights in a depth-$(2d + 1)$ threshold circuit of 'small' weights, *i.e.* $LT_d \subseteq \widehat{LT}_{2d+1}$.

As another application of harmonic analysis, we also show that the *COMPARISON* and *ADDITION* of two $n$-bit numbers is computable with only two layers of elements with 'small' weights, while it was only known to be computable in 3 layers [5]. We also indicate how our 'depth'-efficient algorithm can be applied to show that the product of two $n$-bit numbers can be computed in $\widehat{LT}_4$. In addition, we show that the *MAXIMUM* and *SORTING* of $n$ $n$-bit numbers can be computed in $\widehat{LT}_3$ and $\widehat{LT}_4$, respectively.

## 2    Main Results

**Definition:** Let $X = (x_1, \ldots, x_n)$, $Y = (y_1, \ldots, y_n) \in \{1, -1\}^n$. We consider $X$ and $Y$ as two $n$-bit numbers representing $\sum_{i=1}^n x_i \cdot 2^i$ and $\sum_{i=1}^n y_i \cdot 2^i$, respectively.

The COMPARISON function is defined as

$$C(X, Y) = 1 \text{ iff } X \geq Y$$

In other words,

$$C(X, Y) = sgn\{\sum_{i=1}^n 2^i(x_i - y_i) + 1\}$$

**Lemma 1**

$$COMPARISON \notin \widehat{LT}_1$$

On the other hand, using harmonic analysis [2], we can show the following:

**Lemma 2**

$$COMPARISON \in \widehat{LT}_2$$

**Spectral representation of Boolean functions:** Recently, harmonic analysis has been found to be a powerful tool in studying the *computational complexity* of Boolean functions [1, 2, 7]. The idea is that every Boolean function $f : \{1, -1\}^n \to \{1, -1\}$ can be represented as a polynomial over the field of rational numbers as follows:

$$f(X) = \sum_{\alpha \in \{0,1\}^n} a_\alpha X^\alpha$$

where   $X^\alpha = x_1^{\alpha_1} x_2^{\alpha_2} \ldots x_n^{\alpha_n}$.

Such representation is unique and the coefficients of the polynomial, $\{a_\alpha | \alpha \in \{0,1\}^n\}$, are called the *spectral coefficients* of $f$.

We shall define the $L_1$ *spectral norm* of $f$ to be

$$\|f\| = \sum_{\alpha \in \{0,1\}^n} |a_\alpha|.$$

The proof of Lemma 2 is based on the spectral techniques developed in [2]. Using probabilistic arguments, it was proved in [2] that if a Boolean function has $L_1$ spectral norm which is polynomially bounded, then the function is computable in $\widehat{LT}_2$. We observe (together with Noga Alon) that the techniques in [2] can be generalized to show that any Boolean function with polynomially bounded $L_1$ spectral norm can even be closely *approximated by a sparse polynomial*. This observation is crucial when we extend our result from a single element to networks of elements with large weights.

**Lemma 3** *Let $f(X) : \{1,-1\}^n \to \{1,-1\}$ such that $\|f\| \leq n^c$ for some $c$. Then for any $k > 0$, there exists a sparse polynomial*

$$F(X) = \frac{1}{N} \sum_{\alpha \in S} w_\alpha X^\alpha \text{ such that}$$

$$|F(X) - f(X)| \leq n^{-k},$$

*where $w_\alpha$ and $N$ are integers, $S \subset \{0,1\}^n$, the size of $S$, $w_\alpha$ and $N$ are all bounded by a polynomial in $n$. Hence, $f(X) \in \widehat{LT}_2$.*

As a consequence of this result, Lemma 2 follows since it can be shown that $COMPARISON$ has a polynomially bounded $L_1$ spectral norm.

Now we are ready to state our main results. Although most linear threshold functions require exponentially large weights, we can always simulate them by 3 layers of $\widehat{LT}_1$ elements.

**Theorem 1**

$$LT_1 \subsetneq \widehat{LT}_3$$

The result stated in Theorem 1 implies that a depth-$d$ threshold circuit with exponentially large weights can be simulated by a depth-$3d$ threshold circuit with polynomially large weights. Using the result of Lemma 3, we can actually obtain a more depth-efficient simulation .

**Theorem 2**

$$LT_d \subseteq \widehat{LT}_{2d+1}$$

As another consequence of Lemma 3, we have the following :

**Corollary 1** *Let $f_1(X), ..., f_m(X)$ be functions with polynomially bounded $L_1$ spectral norms, and $g(f_1(X), ..., f_m(X))$ be an $\widehat{LT}_1$ function with $f_i(X)$'s as inputs, i.e.*

$$g(f_1(X), ..., f_m(X)) = sgn(\sum_{i=1}^{m} w_i f_i(X) + w_0)$$

*Then $g$ can be expressed as a sign of a sparse polynomial in $X$ with polynomially many number of monomial terms $X^{\alpha}$'s and polynomially bounded integer coefficients. Hence $g \in \widehat{LT}_2$.*

If all $LT_1$ functions have polynomially bounded $L_1$ spectral norms, then it would follow that $LT_1 \subset \widehat{LT}_2$. However, even the simple MAJORITY function does not have a polynomially bounded $L_1$ spectral norm. We shall prove this fact via the following theorem. (As in Lemma 3, by a sparse polynomial we mean a polynomial with only polynomially many monomial terms $X^{\alpha}$'s).

**Theorem 3** *The $\widehat{LT}_1$ function MAJORITY:*

$$sgn(\sum_{i=1}^{n} x_i)$$

*cannot be approximated by a sparse polynomial with an error $o(n^{-1})$.*

Other applications of the harmonic analysis techniques and the results of Lemma 3 yields the following theorems:

**Theorem 4**

*Let $x, y$ be two $n$-bit numbers. Then*
$$ADDITION(x, y) \in \widehat{LT_2}$$

**Theorem 5** *The product of two $n$-bit integers can be computed in $\widehat{LT}_4$.*

**Theorem 6** *The MAXIMUM of $n$ $n$-bit numbers can be computed in $\widehat{LT}_3$.*

**Theorem 7** *The SORTING of $n$ $n$-bit numbers can be computed in $\widehat{LT}_4$.*

## 3    Concluding Remarks

Our main result indicates that for networks of linear threshold elements, we can trade-off arbitrary real weights with polynomially bounded integer weights, at the expense of a polynomial increase in the size and a factor of almost two in the depth of the network. The proofs of the results in this paper can be found in [13]. We would like to mention that our results have recently been improved by Goldmann, Hastad and Razborov [4]. They showed that any polynomial-size depth-$d$ network of linear threshold elements with arbitrary weights can be simulated by a polynomial-size depth-$(d+1)$ network with "small" (polynomially bounded integer) weights. While our construction can be made explicit, only the existence of the simulation result is proved in [4]; it is left as an open problem in [4] if there is an explicit construction of their results.

## Acknowledgements

This work was done while Kai-Yeung Siu was a research student associate at IBM Almaden Research Center and was supported in part by the Joint Services Program at Stanford University (US Army, US Navy, US Air Force) under Contract DAAL03-88-C-0011, and the Department of the Navy (NAVELEX), NASA Headquarters, Center for Aeronautics and Space Information Sciences under Grant NAGW-419-S6.

## References

[1] J. Bruck. Harmonic Analysis of Polynomial Threshold Functions. *SIAM Journal on Discrete Mathematics*, May 1990.

[2] J. Bruck and R. Smolensky. Polynomial Threshold Functions, $AC^0$ Functions and Spectral Norms. Technical Report RJ 7140, IBM Research, November 1989. Appeared in IEEE Symp. on Found. of Comp. Sci. October, 1990.

[3] A. K. Chandra, L. Stockmeyer, and U. Vishkin. Constant depth reducibility. *Siam J. Comput.*, 13:423–439, 1984.

[4] M. Goldmann, J. Hastad, and A. Razborov  Majority Gates vs. General Weighted Threshold Gates. Unpublished Manuscript.

[5] A. Hajnal, W. Maass, P. Pudlak, M. Szegedy, and G. Turan. Threshold circuits of bounded depth. *IEEE Symp. Found. Comp. Sci.*, 28:99–110, 1987.

[6] R. J. Lechner. Harmonic analysis of switching functions. In A. Mukhopadhyay, editor, *Recent Development in Switching Theory*. Academic Press, 1971.

[7] N. Linial, Y. Mansour, and N. Nisan. Constant Depth Circuits, Fourier Transforms, and Learnability. *Proc. 30th IEEE Symp. Found. Comp. Sci.*, 1989.

[8] S. Muroga and I. Toda. Lower Bound of the Number of Threshold Functions. *IEEE Trans. on Electronic Computers*, EC 15, 1966.

[9] J. Myhill and W. H. Kautz. On the Size of Weights Required for Linear-Input Switching Functions. *IRE Trans. on Electronic Computers*, EC 10, 1961.

[10] I. Parberry and G. Schnitger. Parallel Computation with Threshold Functions . *Journal of Computer and System Sciences*, 36(3):278–302, 1988.

[11] N. Pippenger. The complexity of computations by networks. *IBM J. Res. Develop.*, 31(2), March 1987.

[12] P. Raghavan. Learning in Threshold Networks: A Computation Model and Applications. Technical Report RC 13859, IBM Research, July 1988.

[13] K.-Y. Siu and J. Bruck.  On the Power of Threshold Circuits with Small Weights . *SIAM J. Discrete Math.*, 4(3):423–435, August 1991.

[14] D. R. Smith. Bounds on the Number of Threshold Functions. *IEEE Trans. on Electronic Computers*, EC 15, 1966.

[15] S. Yajima and T. Ibaraki. A Lower Bound on the Number of Threshold Functions. *IEEE Trans. on Electronic Computers*, EC 14, 1965.
